# Waveform Driven Plasticity in BiFeO3 Memristive Devices: Model and Implementation

**Christian Mayr, Paul Staerke, Johannes Partzsch, Rene Schueffny**
Institute of Circuits and Systems
TU Dresden, Dresden, Germany
`{christian.mayr,johannes.partzsch,rene.schueffny}@tu-dresden.de`

**Love Cederstroem**
Zentrum Mikroelektronik Dresden AG
Dresden, Germany
`love.cederstroem@zmdi.com`

**Yao Shuai**
Inst. of Ion Beam Physics and Materials Res.
Helmholtz-Zentrum Dresden-Rossendorf e.V.
Dresden, Germany
`y.shuai@hzdr.de`

**Nan Du, Heidemarie Schmidt**
Professur Materialsysteme der Nanoelektronik
TU Chemnitz, Chemnitz, Germany
`nan.du@s2012.tu-chemnitz.de,Heidemarie.Schmidt@etit.tu-chemnitz.de`

## Abstract

Memristive devices have recently been proposed as efficient implementations of plastic synapses in neuromorphic systems. The plasticity in these memristive devices, i.e. their resistance change, is defined by the applied waveforms. This behavior resembles biological synapses, whose plasticity is also triggered by mechanisms that are determined by local waveforms. However, learning in memristive devices has so far been approached mostly on a pragmatic technological level. The focus seems to be on finding any waveform that achieves spike-timing-dependent plasticity (STDP), without regard to the biological veracity of said waveforms or to further important forms of plasticity. Bridging this gap, we make use of a plasticity model driven by neuron waveforms that explains a large number of experimental observations and adapt it to the characteristics of the recently introduced $BiFeO_3$ memristive material. Based on this approach, we show STDP for the first time for this material, with learning window replication superior to previous memristor-based STDP implementations. We also demonstrate in measurements that it is possible to overlay short and long term plasticity at a memristive device in the form of the well-known triplet plasticity. To the best of our knowledge, this is the first implementations of triplet plasticity on any physical memristive device.

## 1 Introduction

Neuromorphic systems try to replicate cognitive processing functions in integrated circuits. Their complexity/size is largely determined by the synapse implementation, as synapses are significantly more numerous than neurons [1]. With the recent push towards larger neuromorphic systems and higher integration density of these systems, this has resulted in novel approaches especially for the synapse realization. Proposed solutions on the one hand employ nanoscale devices in conjuction with conventional circuits [1] and on the other hand try to integrate as much synaptic functionality (short- and long term plasticity, pulse shaping, etc) in as small a number of devices as possible. In

this context, memristive devices [1] as introduced by L. Chua [2] have recently been proposed as efficient implementations of plastic synapses in neuromorphic systems. Memristive devices offer the possibility of having the actual learning mechanism, synaptic weight storage and synaptic weight effect (i.e. amplification of the presynaptic current) all in one device, compared to the distributed mechanisms in conventional circuit implementations [3]. Moreover, a high-density passive array on top of a conventional semiconductor chip is possible [1]. The plasticity in these memristors, i.e. their resistance change, is defined by the applied waveforms [4], which are fed into the rows and columns of the memristive array by CMOS pre- and postsynaptic neurons [1]. This resembles biological synapses, whose plasticity is also triggered by mechanisms that are determined by local waveforms [5, 6]. However, learning in memristors has so far been approached mostly on a pragmatic technological level. The goal seems to be to find any waveform that achieves spike-timing-dependent plasticity (STDP) [4], without regard to the biological veracity of said waveforms or to further important forms of plasticity [7].

Bridging this gap, we make use of a plasticity rule introduced by Mayr and Partzsch [6] which is driven in a biologically realistic way by neuron waveforms and which explains a large number of experimental observations. We adapt it to a model of the recently introduced $BiFeO_3$ memristive material [8]. Measurement results of the modified plasticity rule implemented on a sample device are given, exhbiting configurable STDP behaviour and pulse triplet [7] reproduction.

## 2 Materials and Methods

### 2.1 Local Correlation Plasticity (LCP)

The LCP rule as introduced by Mayr and Partzsch [6] combines two local waveforms, the synaptic conductance $g(t)$ and the membrane potential $u(t)$. Presynaptic activity is encoded in $g(t)$, which determines the conductance change due to presynaptic spiking. Postsynaptic activity in turn is signaled to the synapse by $u(t)$. The LCP rule combines both in a formulation for the change of the synaptic weight $w$ that is similar to the well-known Bienenstock-Cooper-Munroe rule [9]:

$$\frac{\mathrm{d}w}{\mathrm{d}t} = B \cdot g(t) \cdot (u(t) - \Theta_u) \tag{1}$$

In this equation, $\Theta_u$ denotes the voltage threshold between weight potentiation and depression, which is normally set to the resting potential. Please note that coincident pre- and postsynaptic activities are detected in this rule by multiplication: A weight change only occurs if both presynaptic conductance is elevated and postsynaptic membrane potential is away from rest.

The waveforms for $g(t)$ and $u(t)$ are determined by the employed neuron model. Mayr et al. [6] use a spike response model [10], with waveforms triggered at times of pre- and postsynaptic spikes:

$$g(t) = \hat{G} \cdot e^{-\frac{t-t_n^{\mathrm{pre}}}{\tau_{\mathrm{pre}}}} \qquad \text{for } t_n^{\mathrm{pre}} \leq t < t_{n+1}^{\mathrm{pre}} , \tag{2}$$

$$u(t) = U_{\mathrm{p},n} \cdot \delta(t - t_n^{\mathrm{post}}) + U_{\mathrm{refr}} \cdot e^{-\frac{t-t_n^{\mathrm{post}}}{\tau_{\mathrm{post}}}} \qquad \text{for } t_n^{\mathrm{post}} \leq t < t_{n+1}^{\mathrm{post}} , \tag{3}$$

where $t_n^{\mathrm{pre}}$ and $t_n^{\mathrm{post}}$ denote the $n$-th pre- and postsynaptic spike, respectively. The presynaptic conductance waveform is an exponential with height $\hat{G}$ and decay time constant $\tau_{\mathrm{pre}}$. The postsynaptic potential at a spike is defined by a Dirac pulse with integral $U_{\mathrm{p},n}$, followed by an exponential decay with height $U_{\mathrm{refr}}(< 0)$ and membrane time constant $\tau_{\mathrm{post}}$.

Following [6], postsynaptic adaptation is realised in the value of $U_{\mathrm{p},n}$. For this, $U_{\mathrm{p},n}$ is decreased from a nominal value $U_{\mathrm{p}}$ if the postsynaptic pulse occurs shortly after another postsynaptic pulse:

$$U_{\mathrm{p},n} = U_{\mathrm{p}} \cdot \left(1 - e^{-\frac{t_n^{\mathrm{post}} - t_{n-1}^{\mathrm{post}}}{\tau_{\mathrm{post}}}}\right) \tag{4}$$

The time constant for the exponential decay in this equation is the same as the membrane time constant.

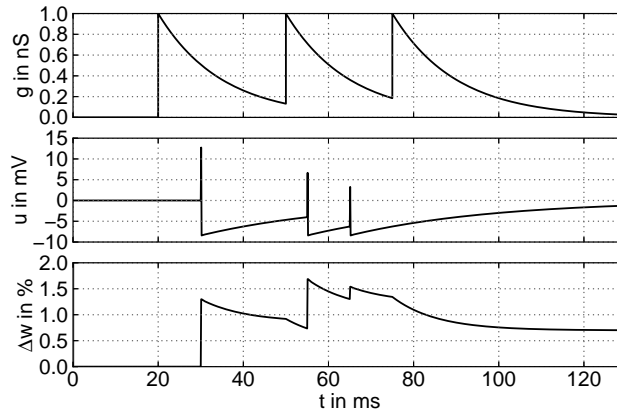

Figure 1: Progression of the conductance $g$, the membrane potential $u$ and the synapse weight $w$ for a sample spike pattern.

Figure 1 shows the pre- and postsynaptic waveforms, as well as the synaptic weight for a sample spike train. For the simple waveforms, two principal weight change mechanisms are present: If the presynaptic side is active at a postsynaptic spike, the weight is instantaneously increased by the large elevation of the membrane potential. In contrast, all presynaptic activity falling into the refractoriness period of the neuron (exponential decay after spike) integrates as a weight decrease.

As shown in [6], this simple model can replicate a multitude of experimental evidence, on par with the most advanced (and complex) phenomenological plasticity models currently available. In addition, the LCP rule directly links synaptic plasticity to other pre- and postsynaptic adaptation processes by their influence on the local waveforms. This can be used to explain further experimental results [6]. In Sec. 3.1, we will adapt the above rule equations to the characteristics of our memristive device, which is introduced in the next section.

## 2.2 Memristive Device

Non-volatile passive analog memory has often been discussed for applications in neuromorphic systems because of the space limitations of analog circuitry. However, until recently only a few groups had access to sufficient materials and devices. Developments in the field of nano material science, especially in the last decade, opened new possibilities for creating compact circuit elements with unique properties.

Most notably after HP released information about their so-called Memristor [11] much effort has been put in the analysis of thin film semiconductor-metal-metaloxide compounds. One of the commonly used materials in this class is $BiFeO_3$ (BFO). The complete conducting mechanisms in BFO are not fully understood yet, with partly contradictory results reported in literature, but it has been confirmed that different physical effects are overlayed and dominate in different states. Particularly the resistive switching effect seems promising for neuromorphic devices and will be discussed in more detail. It has been shown in [12, 8] that the effect can appear uni- or bipolar and is highly dependent on the processing regarding the substrate, growth method, doping, etc. [13].

We use BFO grown by pulsed laser deposition on $Pt/Ti/SiO_2/Si$ substrate with an Au top contact, see in Fig. 2. Memristors were fabricated with circular top plates, which were contacted with needle probes, whereas the continuous bottom plate was contacted at one edge of the die. The BFO films have a thickness of some 100nm. The created devices show a unipolar resistive switching with a rectifying behavior. For a positive bias the device goes into a low resistive state (LRS) and stays there until a negative bias is applied which resets it back to a high resistive state (HRS). The state can be measured without influencing it by applying a low voltage of under 2V.

Figure 3 shows a voltage-current-diagram which indicates some of the characteristics of the device. The measurement consists of three parts: 1) A rising negative voltage is applied which resets the device from an intermediate level to HRS. 2) A rising voltage lowers the resistance exponentially.

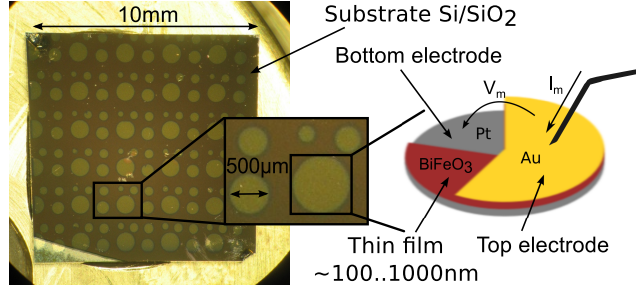

Figure 2: Photograph of the fabricated memristive material that was used for the measurements.

3) A falling positive voltage does not affect the resistance anymore and the relation is nearly ohmic. Because of the rectifying characteristic the current in LRS and HRS for negative voltages does not exhibit as large a dynamic range as for positive voltages.

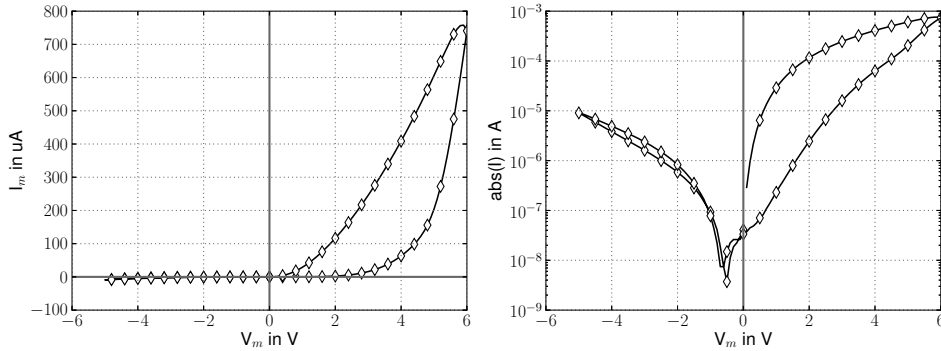

Figure 3: Voltage-current diagram of the device as linear and log-scale plot

## 2.3 Phenomenological Device Model

To apply the LCP model to the BFO device and enable circuit design, a simplified device model is required. We have based our model on the framework of Chua and Kang [2]; that is, using an output function (i.e., for current $I_m$) dependent on time, state and input (i.e., voltage $V_m$). Recently, this has been widely used for the modeling of memristive devices [11, 14, 15]. In contrast to many memristive device models which are based on a sinh function for the output relationship (following Yang et al. [14]), we model the BFO device as two semiconductor junctions. The junctions can abstractly be described by a diode equation: $I_d = I_0(\exp(qV/kT) - 1)$ [16]. In an attempt to catch the basic characteristics, our device could be modeled employing two diode equations letting a state variable, $x$, influence the output and roughly represent the conductance:

$$I_m = h(x, V_m, t) = \left( I_{01} \cdot (e^{d_1 \cdot V_m(t)} - 1) - I_{02} \cdot (e^{-d_2 \cdot V_m(t)} - 1) \right) \cdot x(t) \tag{5}$$

where $V_m$ is the voltage over the device[2] and the diode like equations guarantee a zero crossing hysteresis. The use of parameters $I_{0i}$ and $d_i$ now allows individual control of current characteristics for negative and positive voltages, and as shown in the previous section these are rather asymmetric for our BFO devices. For the purpose of modeling plasticity, our focus has been on the dynamic behavior of the conductance change; this was investigated in some detail by Querlioz et al. [15] and has served as the basis for our model of the state variable:

$$\frac{\mathrm{d}x}{\mathrm{d}t} = f(x, V_m, t) = \Gamma(x) \cdot \Psi(V_m) \tag{6a}$$

In the above the functions $\Gamma(x)$ and $\Psi(V_m)$ relate to how the current state affects the state development and the effect of the applied voltage, respectively. $\Gamma(x)$ is described by an exponential function.

$$\Gamma(x) = \begin{cases} e^{-\beta_1 \frac{x - G_{\min}}{G_{\max} - G_{\min}}}, & V_m(t) > 0, \\ e^{-\beta_2 \frac{G_{\max} - x}{G_{\max} - G_{\min}}}, & V_m(t) \le 0, \ x > G_{\min}, \\ 0, & \text{else} \end{cases} \tag{6b}$$

In $\Psi(V_m)$ we again favor using separate exponential over sinh functions for increased controllability of the different voltage domains (positive and negative). Here the parameters $\varphi_1$ and $\varphi_2$ govern the voltage dependence of the state modification, with $\alpha_1$ and $\alpha_2$ scaling the result. With $\beta_1$ and $\beta_2$, the speed of state saturation is set:

$$\Psi(V_m) = \begin{cases} \alpha_1 \cdot \left( e^{\varphi_1 V_m} - 1 \right), & V_m(t) \ge 0, \\ \alpha_2 \cdot \left( 1 - e^{-\varphi_2 V_m} \right), & V_m(t) < 0, \end{cases} \tag{6c}$$

For implementation, we have used one of the most prominent commercially available simulators for custom analog and mixed-signal integrated circuit design, the Cadence® Spectre®. Using behavioral current sources, the equations for $h(x, V_m, t)$ and $f(x, V_m, t)$ can be implemented and simulated with feasibility for circuit design. Depicted in Fig. 4 are the conductance change over time, at different voltages, for model (Fig. 4a) and measurements (Fig. 4b). It can be seen how the exponential dependency on device voltage gives rise to different levels of operation (Equations (5) and (6c)). Also the saturation of conductance change for a given voltage is visible (Equation (6b)). The sharp changes of current seen in the model are a result of our simplistic approach, whereas the real devices show slower transitions. In addition, it can be noted that above 5 V the real device appears to experience a significantly steeper rise in current. However, the target is to have reasonable characteristics in the region of operation below 5 V which is relevant in our plasticity rule experiments.

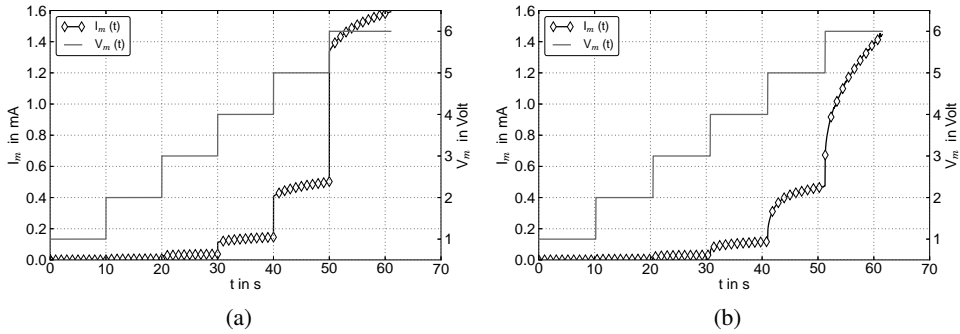

|     |     |
| :-: | :-: |
| (a) | (b) |

Figure 4: Device current for different applied voltages for model (a) and measurement (b).

## 3 Results

### 3.1 Modified LCP

A nonlinearity or learning threshold is required in order to carry out the correlation operation between pre- and postsynaptic waveforms that characterizes various forms of long term learning [9, 17]. In the original LCP rule, this is done by the multiplication of pre- and postsynaptic waveforms, i.e. only coincident activity results in learning. Memristive devices are usually operated in an additive manner, i.e. the pre- and postsynaptic waveforms are applied to both terminals of the device, thus adding/subtracting their voltage curves. In order for the state of the memristive device to only be affected by an overlap of both waveforms, a positive and negative modification threshold is required [4]. As can be seen from equation 6c, the internal voltage driven state change $\Psi(V_m)$ is affected by two different parameters $\varphi_1$ and $\varphi_2$ which govern the thresholds for negative and positive voltages. For our devices, these work out to effective modification thresholds of -2V and

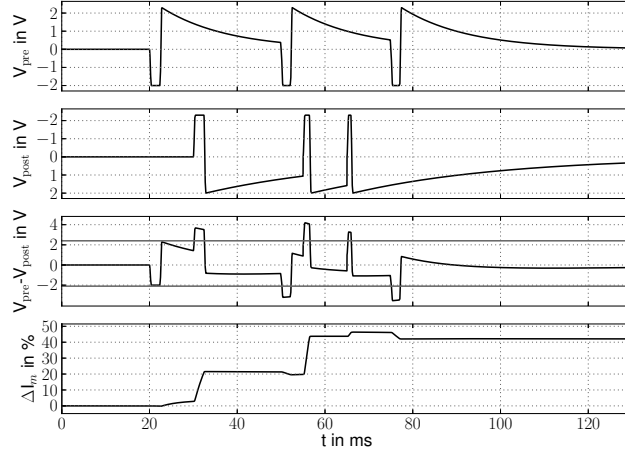

Figure 5: Modification of the original LCP rule for the BFO memristive device, from top to bottom: pre- and postsynaptic voltages/waveforms, exponential decay with $\tau_{pre}$ resp. $\tau_{post}$ (postsynaptic waveform plotted as inverse to illustrate waveform function); resultant voltage difference across memristive device and corresponding memristance modification thresholds (horizontal grey lines); and memristance change as computed from the model of sec. 2.3

+2.3V. Thus, we need waveforms where coincident activity causes a voltage rise above the positive threshold resp. a voltage drop below the negative threshold. In addition, we need a dependence between voltage level and weight change, as the simplest method to differentiate between weights is the voltage saturation characteristic in Fig. 3. That is, a single stimulus (e.g. pulse pairing in STDP) should result in a distinctive memristive programming voltage, driving the memristive device into the corresponding voltage saturation level via the (for typical experiments) 60 stimulus repetitions.

Apart from quantitative adjustments to the original LCP rule, this requires one qualitative adjustment. The presynaptic conductance waveform is now taken as a voltage trace and a short rectangular pulse is added immediately before the exponential downward trace, arriving at a waveform similar to the spike response model for the postsynaptic trace, see uppermost curve in Fig. 5. We call this the modified LCP rule. For overlapping pre- and postsynaptic waveforms, the rectangular pulses of both waveforms 'ride up' on the exponential slopes of their counterparts when looking at the voltage difference $V_m = V_{pre} - V_{post}$ across the memristive device for pre- and postsynaptic waveforms applied to both terminals of the device (see third curve from top in Fig. 5). Since the rectangular pulses are short compared to the exponential waveforms, they represent a constant voltage whose amplitude depends on the time difference between both waveforms (as expressed by the exponential slopes) as required above. Thus, as in the original LCP rule, the exponential slopes of pre- and postsynaptic neuron govern the STDP time windows. Repeated application of such a pre-post pairing drives the memristive device in its corresponding voltage-dependent saturation level.

Similar to the original LCP rule, short term plasticity of the postsynaptic action potentials can now be added to make the model more biologically realistic (e.g. with respect to the triplet learning protocol [6]). We employ the same attenuation function as in equation 4, adjusting the duration of the postsynaptic action potential, see second curve from top in Fig. 5.

Please note: One further important advantage of using this modified LCP rule is that both pre- and postsynaptic waveform are causal, i.e. they start only at the pre- respectively postsynaptic pulse. This is in contrast to most currently proposed waveforms for memristive learning, i.e. these waveforms have to start well in advance of the actual pulse [4], which requires preknowledge of a pulse occurrence. Especially in an unsupervised learning context with self-driven neuron spiking, this preknowledge is simply not existent.

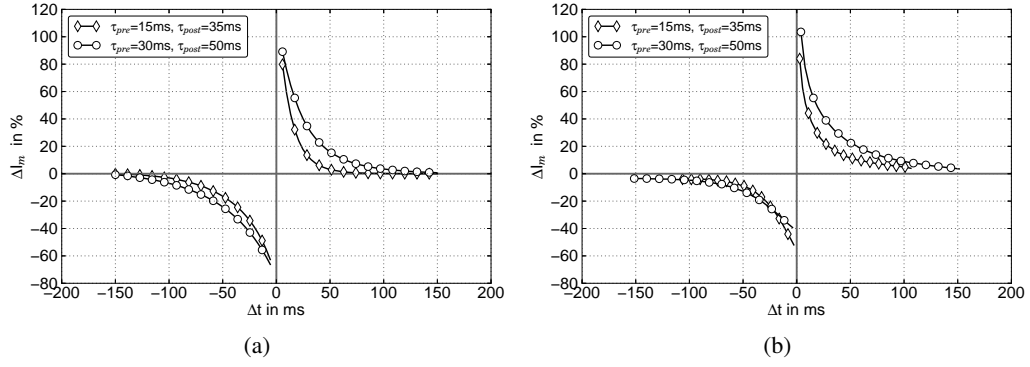

Figure 6: Results for STDP protocol: (a) model simulation, (b) measurement with BFO memristive device.

## 3.2 Measurement results

The waveforms developed in the previous section can be tested in actual protocols for synaptic plasticity. As a first step, we investigate the behaviour of the BFO memristive device in a standard pair-based STDP experiment. For this, we apply 60 spike pairings of different relative timings at a low repetition frequency (4Hz), comparable to biological measurement protocols [17]. Measurements were performed with a BFO memristive device as shown in Fig. 2. As shown in the model simulations in Fig. 6a, the developed waveforms are transformed by the memristive device into approx. exponentially decaying conductance changes. This is in good agreement with biological measurements [17] and common STDP models [7]. The model results are confirmed in measurements for the BFO memristive device, as shown in Fig. 6b. Notably, the measurements result in smooth, continuous curves. This is an expression of the continuous resistance change in the BFO material, which results in a large number of stable resistance levels. This is in contrast e.g. to memristive materials that rely on ferroelectric switching, which exhibit a limited number of discrete resistance levels [18, 1]. Moreover, the nonlinear behaviour of the BFO memristive device has only limited effect on the resulting STDP learning window. The resistance change is directly linked to the applied waveforms. For example, as shown in Fig. 6, an increase in time constants results in correspondingly longer STDP time windows. Following our modeling approach, these time constants are directly linked to the time constants of the underlying neuron and synapse model.

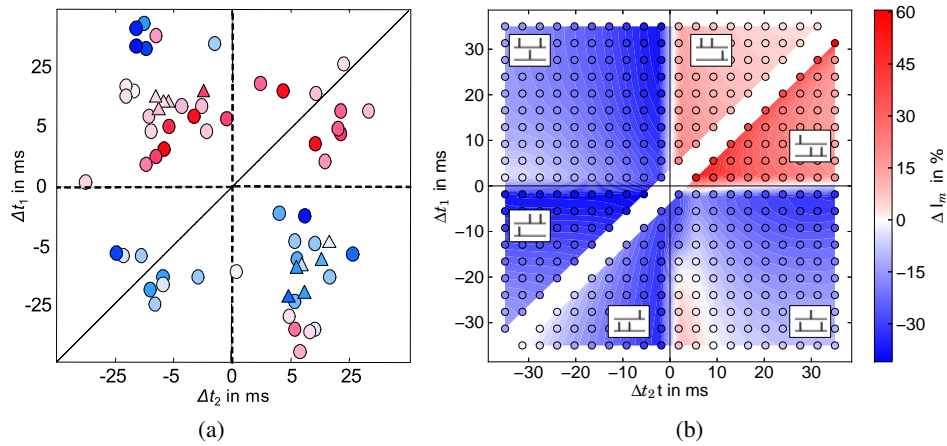

Figure 7: Measurement results for the triplet protocol of Froemke and Dan [7]. (a) biological measurement data, adapted from [7], (b) measurement with BFO memristive device.

Experiments have shown that weight changes of single spike pairings, as expressed by STDP, are nonlinearly integrated when occuring shortly after one another. Commonly, triplets of spikes are used to investigate this effect, as carried out by [7]. The main deviation of these experimental results compared to a pure STDP rule occur for the post-pre-post triplet [6], which can be attributed to postsynaptic adaptation [7]. With this adaptation included in our waveforms (equation 4, as seen in the action potential duration in the second curve from the top of Fig. 5), the BFO memristive device measurements well resemble the post-pre-post results of [7]. The measurement results in Fig. 7b show more depression than the biological data for the pre-post-pre triplet (upper left quadrant). This is because changes in resistance need some time to build up after a stimulating pulse. In the pre-post-pre case, the weight increase has not fully developed when it is overwritten by the second presynaptic pulse, which results in weight decrease. This effect is dependent on the measured device and the parameters of the stimulation waveforms (cf. Supplementary Material).

For keeping the stimulation waveforms as simple as possible, only postsynaptic adaptation has been included. However, it has been shown that presynaptic short-term plasticity also has a strong influence on long-term learning [19, 6]. With our modeling approach, a model of short-term plasticity can be easily connected to the stimulation waveforms by modulating the length of the presynaptic pulse. Along the same lines, the postsynaptic waveform can be shifted by a slowly changing voltage analogous to the original LCP rule (cf. Eq. 1) to introduce a metaplastic regulation of weight potentiation and depression [6]. Together, these extensions open up an avenue for the seamless integration of different forms of plasticity in learning memristive devices.

## 3.3 Conclusion

Starting from a waveform-based general plasticity rule and a model of the memristive device, we have shown a direct way to go from these premises to biologically realistic learning in a $BiFeO_3$ memristive device. Employing the LCP rule for memristive learning has several advantages. As a memristor is a two-terminal device, the separation of the learning in two waveforms in the LCP rule lends itself naturally to employing it in a passive array of memristors [1, 4]. In addition, this waveform-defined plasticity behaviour enables easy control of the STDP time windows, which is further aided by the excellent multi-level memristive programming capability of the $BiFeO_3$ memristive devices. There is only a very small number of memristors where plasticity has been shown at actual devices at all [18, 1]. Among those, our highly-configurable, finely grained learning curves are unique, other implementations exhibit statistical variations [1], can only assume a few discrete levels [18] or the learning windows are device-inherent, i.e. cannot be adjusted [20]. This comes at the price that in contrast to e.g. phase-change materials, $BiFeO_3$ is not easily integrated on top of CMOS [8].

The waveform-defined plasticity of the LCP rule enables the explicit inclusion of short term plasticity in long term memristive learning, as shown for the triplet protocol. As the pre- and postsynaptic waveforms are generated in the CMOS neuron circuits below the memristive array [1], short term plasticity can thus be added at little extra overall circuit cost and without modification of the memristive array itself. In contrast to our easily controlled short term plasticity, the only previous work targeting memristive short term plasticity employed intrinsic (i.e. non-controllable) device properties [20]. To the best of our knowledge, this is the first time triplets or other higher-order forms of plasticity have been shown for a physical memristive device.

In a wider neuroscience context, waveform defined plasticity as shown here could be seen as a general computational principle, i.e. synapses are not likely to measure time differences as in naive forms of STDP rules, they are more likely to react to local static [21] and dynamic [5] state variables. Some interesting predictions could be derived from that, e.g. STDP time constants that are linked to synaptic conductance changes or to the membrane time constant [22, 6]. These predictions could be easily verified experimentally.

## Acknowledgments

The research leading to these results has received funding from the European Union Seventh Framework Programme (FP7/2007- 2013) under grant agreement no. 269459 (Coronet).

## Footnotes

[1] In 1971 Leon Chua postulated the existence of a device where the current or voltage is directly controlled by voltage flux or charge respectively, this was called a *memristor*. Using a general state space description Chua and Kang later extended the theory to cover the very broad class of *memristive devices* [2]. Even though the two terms are used interchangeably in other studies, since the devices used in this study do not fit the strict definition of memristor, we will refer to them as memristive devices in the following.

[2]With $\sinh(z) = 1/2 \cdot (e^z - e^{-z})$, our approach is not fundamentally different from using a sinh function.

# References

[1] S. H. Jo, T. Chang, I. Ebong, B. B. Bhadviya, P. Mazumder, and W. Lu, "Nanoscale memristor device as synapse in neuromorphic systems," *Nano Letters*, vol. 10, no. 4, pp. 1297–1301, 2010.

[2] L. Chua and S. M. Kang, "Memristive devices and systems," *Proceedings of the IEEE*, vol. 64, no. 2, pp. 209 – 223, feb. 1976.

[3] S. Fusi, M. Annunziato, D. Badoni, A. Salamon, and D. Amit, "Spike-driven synaptic plasticity: Theory, simulation, VLSI implementation," *Neural Computation*, vol. 12, pp. 2227–2258, 2000.

[4] M. Laiho, E. Lehtonen, A. Russel, and P. Dudek, "Memristive synapses are becoming reality," *The Neuromorphic Engineer*, November 2010. [Online]. Available: http://www.ine-news.org/view.php?source=003396-2010-11-26

[5] S. Dudek and M. Bear, "Homosynaptic long-term depression in area CAl of hippocampus and effects of N-methyl-D-aspartate receptor blockade," *PNAS*, vol. 89, pp. 4363–4367, 1992.

[6] C. Mayr and J. Partzsch, "Rate and pulse based plasticity governed by local synaptic state variables," *Frontiers in Synaptic Neuroscience*, vol. 2, pp. 1–28, 2010.

[7] R. Froemke and Y. Dan, "Spike-timing-dependent synaptic modification induced by natural spike trains," *Nature*, vol. 416, pp. 433–438, 2002.

[8] Y. Shuai, S. Zhou, D. Burger, M. Helm, and H. Schmidt, "Nonvolatile bipolar resistive switching in au/bifeo[sub 3]/pt," *Journal of Applied Physics*, vol. 109, no. 12, p. 124117, 2011. [Online]. Available: http://link.aip.org/link/?JAP/109/124117/1

[9] E. Bienenstock, L. Cooper, and P. Munro, "Theory for the development of neuron selectivity: orientation specificity and binocular interaction in visual cortex," *Journal of Neuroscience*, vol. 2, pp. 32–48, 1982.

[10] W. Gerstner and W. Kistler, *spiking neuron models: single neurons, populations, plasticity*. Cambridge University Press, 2002.

[11] D. B. Strukov, G. S. Snider, D. R. Stewart, and R. S. Williams, "The missing memristor found," *Nature*, vol. 453, no. 7191, pp. 80–83, May 2008. [Online]. Available: http://dx.doi.org/10.1038/nature06932

[12] C. Wang, K. juan Jin, Z. tang Xu, L. Wang, C. Ge, H. bin Lu, H. zhong Guo, M. He, and G. zhen Yang, "Switchable diode effect and ferroelectric resistive switching in epitaxial bifeo[sub 3] thin films," *Applied Physics Letters*, vol. 98, no. 19, p. 192901, 2011.

[13] Y. Shuai, S. Zhou, C. Wu, W. Zhang, D. Bürger, S. Slesazeck, T. Mikolajick, M. Helm, and H. Schmidt, "Control of rectifying and resistive switching behavior in bifeo₃ thin films," *Applied Physics Express*, vol. 4, no. 9, p. 095802, 2011. [Online]. Available: http://apex.jsap.jp/link?APEX/4/095802/

[14] J. J. AU Yang, M. D. Pickett, X. Li, O. A. A., D. R. Stewart, and R. S. Williams, "Memristive switching mechanism for metal//oxide//metal nanodevices," *Nature Nanotechnology*, pp. 429,430,431,432,433, July 2008. [Online]. Available: http://dx.doi.org/10.1038/nnano.2008.160

[15] D. Querlioz, P. Dollfus, O. Bichler, and C. Gamrat, "Learning with memristive devices: How should we model their behavior?" in *Nanoscale Architectures (NANOARCH), 2011 IEEE/ACM International Symposium on*, june 2011, pp. 150 –156.

[16] B. G. Streetman and S. K. Banerjee, *Solid State Electronic Devices*. Pearson Prentice Hall, 2006.

[17] G.-Q. Bi and M.-M. Poo, "Synaptic modifications in cultured hippocampal neurons: dependence on spike timing, synaptic strength, and postsynaptic cell type," *Journal of Neuroscience*, vol. 18, no. 24, pp. 10 464–10 472, 1998.

[18] F. Alibart, S. Pleutin, O. Bichler, C. Gamrat, T. Serrano-Gotarredona, B. Linares-Barranco, and D. Vuillaume, "A memristive nanoparticle/organic hybrid synapstor for neuroinspired computing," *Advanced Functional Materials*, vol. 22, no. 3, pp. 609–616, 2012. [Online]. Available: http://dx.doi.org/10.1002/adfm.201101935

[19] R. Froemke, I. Tsay, M. Raad, J. Long, and Y. Dan, "Contribution of individual spikes in burst-induced long-term synaptic modification," *Journal of Neurophysiology*, vol. 95, pp. 1620–1629, 2006.

[20] T. Ohno, T. Hasegawa, T. Tsuruoka, K. Terabe, J. Gimzewski, and M. Aono, "Short-term plasticity and long-term potentiation mimicked in single inorganic synapses," *Nature Materials*, vol. 10, pp. 591–595, 2011.

[21] A. Ngezahayo, M. Schachner, and A. Artola, "Synaptic activity modulates the induction of bidirectional synaptic changes in adult mouse hippocampus," *The Journal of Neuroscience*, vol. 20, no. 3, pp. 2451–2458, 2000.

[22] J.-P. Pfister, T. Toyoizumi, D. Barber, and W. Gerstner, "Optimal spike-timing dependent plasticity for precise action potential firing in supervised learning," *Neural Computation*, vol. 18, pp. 1309–1339, 2006.

